# Structured Machine Learning For 'Soft' Classification with Smoothing Spline ANOVA and Stacked Tuning, Testing and Evaluation

**Grace Wahba**
Dept of Statistics
University of Wisconsin
Madison, WI 53706

**Yuedong Wang**
Dept of Statistics
University of Wisconsin
Madison, WI 53706

**Chong Gu**
Dept of Statistics
Purdue University
West Lafayette, IN 47907

**Ronald Klein, MD**
Dept of Ophthalmalogy
University of Wisconsin
Madison, WI 53706

**Barbara Klein, MD**
Dept of Ophthalmalogy
University of Wisconsin
Madison, WI 53706

## Abstract

We describe the use of smoothing spline analysis of variance (SS-ANOVA) in the penalized log likelihood context, for learning (estimating) the probability $p$ of a '1' outcome, given a training set with attribute vectors and outcomes. $p$ is of the form $p(t) = e^{f(t)}/(1 + e^{f(t)})$, where, if $t$ is a vector of attributes, $f$ is learned as a sum of smooth functions of one attribute plus a sum of smooth functions of two attributes, etc. The smoothing parameters governing $f$ are obtained by an iterative unbiased risk or iterative GCV method. Confidence intervals for these estimates are available.

## 1. Introduction to 'soft' classification and the bias-variance tradeoff.

In medical risk factor analysis records of attribute vectors and outcomes (0 or 1) for each example (patient) for $n$ examples are available as training data. Based on the training data, it is desired to estimate the *probability $p$* of the 1 outcome for any

new examples in the future, given their attribute vectors. In 'soft' classification, the estimate $\hat{p}$ of $p$ is of particular interest, and might be used, say, by a physician to tell a patient that if he reduces his cholesterol from $t$ to $t'$, then he will reduce his risk of a heart attack from $\hat{p}(t)$ to $\hat{p}(t')$. We assume here that $p$ varies 'smoothly' with any continuous attribute (predictor variable).

It is long known that smoothness penalties and Bayes estimates are intimately related (see *e.g.* Kimeldorf and Wahba(1970, 1971), Wahba(1990) and references there). Our philosophy with regard to the use of priors in Bayes estimates is to use them to generate families of reasonable estimates (or families of penalty functionals) indexed by those smoothing or regularization parameters which are most relevant to controlling the generalization error. (See Wahba(1990) Chapter 3, also Wahba(1992)). Then use cross-validation, generalized cross validation (GCV), unbiased risk estimation or some other *performance oriented* method to choose these parameter(s) to minimize a computable proxy for the generalization error. A person who believed the relevant prior might use maximum likelihood (ML) to choose the parameters, but ML may not be robust against an unrealistic prior (that is, ML may not do very well from the generalization point of view if the prior is off), see Wahba(1985). One could assign a hyperprior to these parameters. However, except in cases where real prior information is available, there is no reason to believe that the use of hyperpriors will beat out a performance oriented criterion based on a good proxy for the generalization error, assuming, of course, that low generalization error is the true goal.

O'Sullivan *et al*(1986) proposed a penalized log likelihood estimate of $f$, this work was extended to the SS-ANOVA context in Wahba, Gu, Wang and Chappell(1993), where numerous other relevant references are cited. This paper is available by ftp from `ftp.stat.wisc.edu`, `cd pub/wahba` in the file `soft-class.ps.Z`. An extended bibliography is available in the same directory as `ml-bib.ps`. The SS-ANOVA allows a variety of *interpretable* structures for the possible relationships between the predictor variables and the outcome, and reduces to simple relations in some of the attributes, or even, to a two-layer neural net, when the data suggest that such a representation is adequate.

## 2. Soft classification and penalized log likelihood risk factor estimation

To describe our 'worldview', let $t$ be a vector of attributes, $t \in \Omega \in \mathcal{T}$, where $\Omega$ is some region of interest in attribute space $\mathcal{T}$. Our 'world' consists of an arbitrarily large population of potential examples, whose attribute vectors are distributed in some way over $\Omega$ and, considering all members of this 'world' with attribute vectors in a small neighborhood about $t$, the fraction of them that are 1's is $p(t)$. Our training set is assumed to be a random sample of $n$ examples from this population, whose outcomes are known, and our goal is to estimate $p(t)$ for any $t \in \Omega$. In 'soft' classification, we do not expect one outcome or the other to be a 'sure thing', that is we do not expect $p(t)$ to be 0 or 1 for large portions of $\Omega$.

Next, we review penalized log likelihood risk estimates. Let the training data be $\{y_i, t(i), i = 1, ...n\}$ where $y_i$ has the value 1 or 0 according to the classification of example $i$, and $t(i)$ is the attribute vector for example $i$. If the $n$ examples are a random sample from our 'world', then the likelihood function of this data, given

$p(\cdot)$, is

$$likelihood\{y,p\} = \Pi_{i=1}^n p(t(i))^{y_i}(1-p(t(i)))^{1-y_i}, \tag{1}$$

which is the product of $n$ Bernoulli likelihoods. Define the logit $f(t)$ by $f(t) = \log[p(t)/(1-p(t))]$, then $p(t) = e^{f(t)}/(1+e^{f(t)})$. Substituting in $f$ and taking logs gives

$$- log\ likelihood\{y,f\} \equiv \mathcal{L}(y,f) = \sum_{i=1}^n \log(1+e^{f(t(i))}) - y_i f(t(i)). \tag{2}$$

We estimate $f$ assuming that it is in some space $\mathcal{H}$ of smooth functions. (Technically, $\mathcal{H}$ is a reproducing kernel Hilbert space, see Wahba(1990), but you don't need to know what this is to read on). The fact that $f$ is assumed 'smooth' makes the methods here very suitable for medical data analysis. The penalized log likelihood estimate $f_\lambda$ of $f$ will be obtained as the minimizer in $\mathcal{H}$ of

$$\mathcal{L}(y,f) + \frac{n}{2}\lambda J(f) \tag{3}$$

where $J(f)$ is a suitable 'smoothness' penalty. A simple example is, $\mathcal{T} = [0,1]$ and $J(f) = \int_0^1 (f^{(m)}(t))^2 dt$, in which case $f_\lambda$ is a polynomial spline of degree $2m-1$. If

$$J_m^d(f) = \sum_{\alpha_1+\cdots+\alpha_d=m} \frac{m!}{\alpha_1!\cdots\alpha_d!} \int_{-\infty}^{\infty}\cdots\int_{-\infty}^{\infty} \left(\frac{\partial^m f}{\partial x_1^{\alpha_1}\cdots\partial x_d^{\alpha_d}}\right)^2 \prod_j dx_j. \tag{4}$$

then $f_\lambda$ is a thin plate spline. The thin plate spline is a linear combination of polynomials of degree $m$ or less in $d$ variables, and certain radial basis functions. For more details and other penalty functionals which result in rbf's, see Wahba(1980, 1990, 1992).

The likelihood function $\mathcal{L}(y,f)$ will be maximized if $p(t(i))$ is 1 or 0 according as $y_i$ is 1 or 0. Thus, in the (full-rank) spline case, as $\lambda \to 0$, $f_\lambda$ tends to $+\infty$ or $-\infty$ at the data points. Therefore, by letting $\lambda$ be small, we can come close to fitting the data points exactly, but unless the 1's and 0's are well separated in attribute space, $f_\lambda$ will be a very 'wiggly' function and the generalization error (not precisely defined yet) may be large.

The choice of $\lambda$ represents a tradeoff between overfitting and underfitting the data (bias-variance tradeoff). It is important in practice good value of $\lambda$. We now define what we mean by a good value of $\lambda$. Given the family $p_\lambda, \lambda \geq 0$, we want to choose $\lambda$ so that $p_\lambda$ is close to the 'true' but unknown $p$ so that, if new examples arrive with attribute vector in a neighborhood of $t$, $p_\lambda(t)$ will be a good estimate of the fraction of them that are 1's. 'Closeness' can be defined in various reasonable ways. We use the Kullbach-Leibler ($KL$) distance (not a real distance!). The $KL$ distance between two probability measures $(g,\hat{g})$ is defined as $KL(g,\hat{g}) = E_g[\log(g/\hat{g})]$, where $E_g$ means expectation given $g$ is the true distribution. If $\nu(t)$ is some probability measure on $\mathcal{T}$, (say, a proxy for the distribution of the attributes in the population), then define $KL_\nu(p,p_\lambda)$ (for Bernoulli random variables) with respect to $\nu$ as

$$KL_\nu(p,p_\lambda) = \int \left[p(t)log\left(\frac{p(t)}{p_\lambda(t)}\right) + (1-p(t))\log\left(\frac{1-p(t)}{1-p_\lambda(t)}\right)\right] d\nu(t). \tag{5}$$

Since $KL_\nu$ is not computable from the data, it is necessary to develop a computable proxy for it. By a computable proxy is meant a function of $\lambda$ that can be calculated from the training set which has the property that its minimizer is a good estimate of the minimizer of $KL_\nu$. By letting $p_\lambda(t) = e^{f_\lambda(t)}/(1 + e^{f_\lambda(t)})$ it is seen that to minimize $KL_\nu$, it is only necessary to minimize

$$\int [log(1 + e^{f_\lambda(t)}) - p(t)f_\lambda(t)]d\nu(t) \tag{6}$$

over $\lambda$ since (5) and (6) differ by something that does not depend on $\lambda$. Leaving-out-half cross validation ($\frac{1}{2}CV$) is one conceptually simple and generally defensible (albeit possibly wasteful) way of choosing $\lambda$ to minimize a proxy for $KL_\nu(p, p_\lambda)$. The $n$ examples are randomly divided in half and the first $n/2$ examples are used to compute $p_\lambda$ for a series of trial values of $\lambda$. Then, the remaining $n/2$ examples are used to compute

$$\widehat{KL}_{\frac{1}{2}CV}(\lambda) = \frac{2}{n} \sum_{i=\frac{n}{2}+1}^{n} [log(1 + e^{f_\lambda(t(i))}) - y_i f_\lambda(t(i))] \tag{7}$$

for the trial values of $\lambda$. Since the expected value of $y_i$ is $p(t(i))$, (7) is, for each $\lambda$ an unbiased estimate of (6) with $d\nu$ the sampling distribution of the $\{t(1), ..., t(n/2)\}$. $\lambda$ would then be chosen by minimizing (7) over the trial values. It is inappropriate to just evaluate (7) using the same data that was used to obtain $f_\lambda$, as that would lead to overfitting the data. Variations on (7) are obtained by successively leaving out groups of data. Leaving-out-one versions of (7) may be defined, but the computation may be prohibitive.

## 3. Newton-Raphson Iteration and the Unbiased Risk estimate of $\lambda$.

We use the unbiased risk estimate given in Craven and Wahba(1979) for smoothing spline estimation with Gaussian errors, which has been adapted by Gu(1992a) for the Bernoulli case. To describe the estimate we need to describe the Newton-Raphson iteration for minimizing (3). Let $b(f) = log(1 + e^f)$, then $\mathcal{L}(y, f) = \sum_{i=1}^{n} [b(f(t(i))) - y_i f(t(i))]$. It is easy to show that $Ey_i = f(t(i)) = b'(f(t(i)))$ and $var\ y_i = p(t(i))(1 - p(t(i))) = b''(f(t(i)))$. Represent $f$ either exactly by using a basis for the (known) $n$-space of functions containing the solution, or approximately by suitable approximating basis functions, to get

$$f \simeq \sum_{k=1}^{N} c_k B_k. \tag{8}$$

Then we need to find $c = (c_1, ..., c_N)'$ to minimize

$$I_\lambda(c) = \sum_{i=1}^{n} b(\sum_{k=1}^{N} c_k B_k(t(i))) - y_i(\sum_{k=1}^{N} c_k B_k(t(i))) + \frac{n}{2}\lambda c'\Sigma c, \tag{9}$$

where $\Sigma$ is the necessarily non-negative definite matrix determined by $J(\sum_k c_k B_k) = c'\Sigma c$. The gradient $\nabla I_\lambda$ and the Hessian $\nabla^2 I_\lambda$ of $I_\lambda$ are given by

$$\nabla I_\lambda = \begin{pmatrix} \frac{\partial I_\lambda}{\partial c_1} \\ \vdots \\ \frac{\partial I_\lambda}{\partial c_N} \end{pmatrix} = X'(p_c - y) + n\lambda\Sigma c, \tag{10}$$

$$\{\nabla^2 I_\lambda\}_{jk} = \frac{\partial^2 I_\lambda}{\partial c_j \partial c_k} = X'W_c X + n\lambda\Sigma, \tag{11}$$

where $X$ is the matrix with $ij$th entry $B_j(t(i))$, $p_c$ is the vector with $i$th entry $p_c(t(i))$ given by $p_c(t(i)) = \frac{e^{f_c(t(i))}}{(1+e^{f_c(t(i))})}$ where $f_c(\cdot) = \sum_{k=1}^N c_k B_k(\cdot)$, and $W_c$ is the diagonal matrix with $ii$th entry $p_c(t(i))(1-p_c(t(i)))$. Given the $\ell$th Newton-Raphson iterate $c^{(\ell)}$, $c^{(\ell+1)}$ is given by

$$c^{(\ell+1)} = c^{(\ell)} - (X'W_{c^{(\ell)}}X + n\lambda\Sigma)^{-1}(X'(p_{c^{(\ell)}} - y) + n\lambda\Sigma c^{(\ell)}) \tag{12}$$

and $c^{(\ell+1)}$ is the minimizer of

$$I_\lambda^{(\ell)}(c) = \|z^{(\ell)} - W_{c^{(\ell)}}^{1/2}Xc\|^2 + n\lambda c'\Sigma c. \tag{13}$$

where $z^{(\ell)}$, the so-called pseudo-data, is given by

$$z^{(\ell)} = W_{c^{(\ell)}}^{-1/2}(y - p_{c^{(\ell)}}) + W_{c^{(\ell)}}^{1/2}Xc^{(\ell)}. \tag{14}$$

The 'predicted' value $\hat{z}^{(\ell)} = W_{c^{(\ell)}}^{1/2}Xc$, where $c$ is the minimizer of (13), is related to the pseudo-data $z^{(\ell)}$ by

$$\hat{z}^{(\ell)} = A^{(\ell)}(\lambda)z^{(\ell)}, \tag{15}$$

where $A^{(\ell)}(\lambda)$ is the smoother matrix given by

$$A^{(\ell)}(\lambda) = W_{c^{(\ell)}}^{1/2}X(X'W_{c^{(\ell)}}X + n\lambda\Sigma)^{-1}X'W_{c^{(\ell)}}^{1/2}. \tag{16}$$

In Wahba(1990), Section 9.2 [1], it was proposed to obtain a GCV score for $\lambda$ in (9) as follows: For fixed $\lambda$, iterate (12) to convergence. Define $V^{(\ell)}(\lambda) = \frac{1}{n}\|(I - A^{(\ell)}(\lambda))z^{(\ell)}\|^2/(\frac{1}{n}tr(I - A^{(\ell)}(\lambda)))^2$. Letting $L$ be the converged value of $\ell$, compute

$$V^{(L)}(\lambda) = \frac{\frac{1}{n}\|(I - A^{(L)}(\lambda))z^{(L)}\|^2}{(\frac{1}{n}tr(I - A^{(L)}(\lambda)))^2} \sim \frac{\frac{1}{n}\|W_{c^{(L)}}^{-1/2}(y - p_{c^{(L)}})\|^2}{(\frac{1}{n}tr(I - A^{(L)}(\lambda)))^2} \tag{17}$$

and minimize $V^{(L)}$ with respect to $\lambda$. Gu(1992a) showed that (since the variance is known once the mean is known here) that the unbiased risk estimate $U(\lambda)$ in Craven and Wahba can also be adapted to this problem as

$$U^{(\ell)}(\lambda) = \frac{1}{n}\|W_{c^{(\ell)}}^{-1/2}(y - p_{c^{(\ell)}})\|^2 + \frac{2}{n}tr\, A^{(\ell)}(\lambda). \tag{18}$$

He also proposed an alternating iteration, different than that described in Wahba(1990), namely, given $c^{(\ell)} = c^{(\ell)}(\lambda^{(\ell)})$, find $\lambda = \lambda^{(\ell+1)}$ to minimize (18). Given $\lambda^{(\ell+1)}$, do a Newton step to get $c^{(\ell+1)}$, get $\lambda^{(\ell+2)}$ by minimizing (18), continue until convergence. He showed that the alternating iteration gave better estimates of $\lambda$ using $V$ than the iteration in Wahba(1990), as measured by the $KL$-distance. His results (with the alternating iteration) suggested $U$ had somewhat of an advantage over $V$, and that is what we are using in the present work. Zhao *et al*, this volume, have used $V$ successfully with the alternating iteration.

## 4. Smoothing spline analysis of variance (SS-ANOVA)

In SS-ANOVA, $f(t) = f(t_1, ..., t_d)$ is decomposed as

$$f(t) = \mu + \sum_\alpha f_\alpha(t_\alpha) + \sum_{\alpha<\beta} f_{\alpha\beta}(t_\alpha, t_\beta) + \cdots \qquad (19)$$

where the terms in the expansion are uniquely determined by side conditions which generalize the side conditions of the usual ANOVA decompositions. Let the logit $f(t)$ be of the form (19) where the terms are summed over $\alpha \in \mathcal{M}, \alpha, \beta \in \mathcal{M}$, etc. where $\mathcal{M}$ indexes terms which are chosen to be retained in the model after a model selection procedure. Then $f_{\lambda,\theta}$, an estimate of $f$, is obtained as the minimizer of

$$\mathcal{L}(y, f_{\lambda,\theta}) + \lambda J_\theta(f) \qquad (20)$$

where

$$J_\theta(f) = \sum_{\alpha\in\mathcal{M}} \theta_\alpha^{-1} J_\alpha(f_\alpha) + \sum_{\alpha,\beta\in\mathcal{M}} \theta_{\alpha\beta}^{-1} J_{\alpha\beta}(f_{\alpha\beta}) + \cdots \qquad (21)$$

The $J_\alpha, J_{\alpha\beta}, \cdots$ are quadratic 'smoothness' penalty functionals, and the $\theta$'s satisfy a single constraint. For certain spline-like smoothness penalties, the minimizer of (20) is known to be in the span of a certain set of $n$ functions, and the vector $c$ of coefficients of these functions can (for fixed $(\lambda, \theta)$) be chosen by the Newton Raphson iteration. Both $\lambda$ and the $\theta$'s are estimated by the unbiased risk estimate of Gu using RKPACK(available from netlib@research.att.com) as a subroutine at each Newton iteration. Details of smoothing spline ANOVA decompositions may be found in Wahba(1990) and in Gu and Wahba(1993) (also available by ftp to ftp.stat.wisc.edu, cd to pub/wahba , in the file ssanova.ps.Z). In Wahba et al(1993) op cit, we estimate the risk of diabetes given some of the attributes in the Pima-Indian data base. There $\mathcal{M}$ was chosen partly by a screening process using paramteric GLIM models and partly by a leaving out approximately 1/3 procedure.

Continuing work involves development of confidence intervals based on Gu(1992b), development of numerical methods suitable for very large data sets based on Girard's(1991) randomized trace estimation, and further model selection issues.

In the Figures we provide some preliminary analyses of data from the Wisconsin Epidemiological Study of Diabetic Retinopathy (WESDR, Klein et al 1988). The data used here is from people with early onset diabetes participating in the WESDR study.    Figure 1(left) gives a plot of body mass index (bmi) (a measure of obesity) vs age (age) for 669 instances (subjects) in the WESDR study that had no diabetic retinopathy or nonproliferative retinopathy at the start of the study. Those subjects who had (progressed) retinopathy four years later, are marked as * and those with no progression are marked as ·. The contours are lines of estimated posterior standard deviation of the estimate $\hat{p}$ of the probability of progression. These contours are used to delineate a region in which $\hat{p}$ is deemed to be reliable. Glycosylated hemoglobin (gly), a measure of blood sugar control. was also used in the estimation of $p$. A model of the form $p = e^f/(1+e^f)$, $f(\text{age}, \text{gly}, \text{bmi}) = \mu + f_1(\text{age}) + b \cdot \text{gly} + f_3(\text{bmi}) + f_{13}(\text{age}, \text{bmi})$ was selected using some of the screening procedures described in Wahba et al(1993), along with an examination of the estimated multiple smoothing parameters, which indicated that the linear term in gly was sufficient to describe the (quite strong) dependence on gly. Figure 1(right) shows the estimated probability of progression

given by this model. Figure 2(left) gives cross sections of the fitted model of Figure 1(right), and Figure 2(right) gives another cross section, along with its confidence interval. Interesting observations can be made, for example, persons in their late 20's with higher **gly** and **bmi** are at greatest risk for progression of the disease.

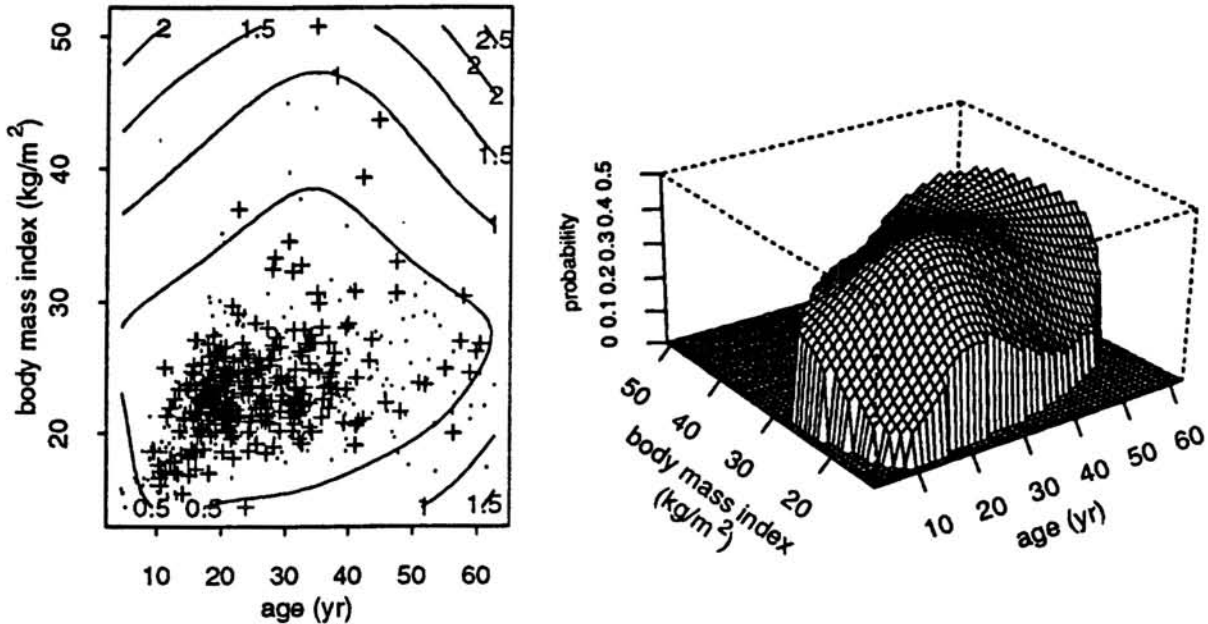

Figure 1: Left: Data and contours of constant posterior standard deviation at the median **gly**, as a function of **age** and **bmi**. Right: Estimated probability of progression at the median **gly**, as a function of **age** and **bmi**.

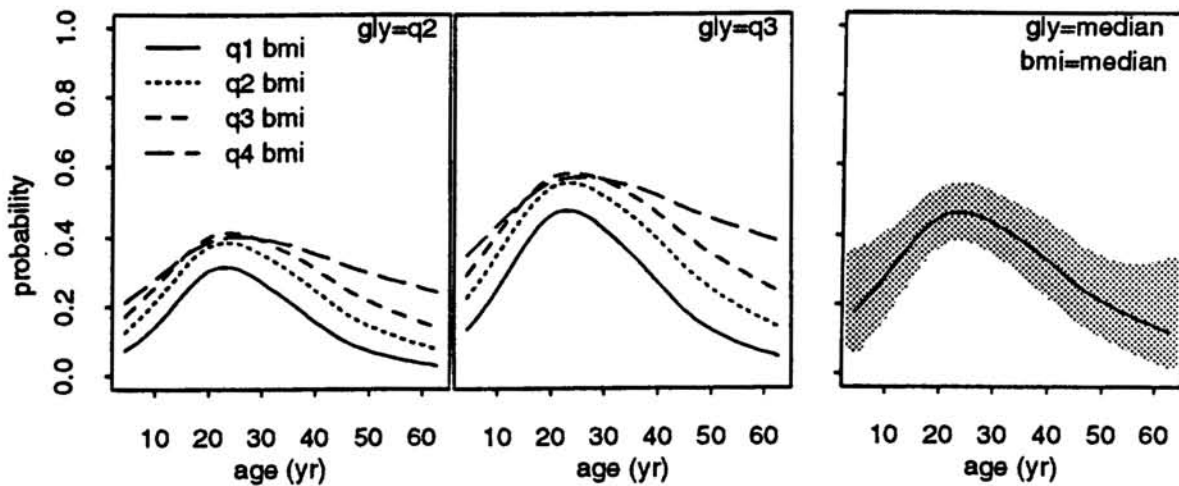

Figure 2: Left: Eight cross sections of the right panel of Figure 1, Estimated probability of progression as a function of **age**, at four levels of **bmi** by two of **gly**. q1,...q4 are the quartiles at .125, .375, .625 and .875. Right: Cross section of the right panel of Figure 1 for **bmi** and **gly** at their medians, as a function of **age**, with Bayesian 'condifidence interval' (shaded) which generalizes Gu(1992b) to the multivariate case.

## Acknowledgements

Supported by NSF DMS-9121003 and DMS-9301511, and NEI-NIH EY09946 and EY03083

## Footnotes

[1] The definition of $\lambda$ there differs from the definition here by a factor of $n/2$. Please note the typographical error in (9.2.18) there where $\lambda$ should be $2\lambda$.

## References

Craven, P. & Wahba, G. (1979), 'Smoothing noisy data with spline functions: estimating the correct degree of smoothing by the method of generalized cross-validation', *Numer. Math.* **31**, 377–403.

Girard, D. (1991), 'Asymptotic optimality of the fast randomized versions of GCV and $C_L$ in ridge regression and regularization', *Ann. Statist.* **19**, 1950–1963.

Gu, C. (1992a), 'Cross-validating non-Gaussian data', *J. Comput. Graph. Stats.* **1**, 169–179.

Gu, C. (1992b), 'Penalized likelihood regression: a Bayesian analysis', *Statistica Sinica* **2**, 255–264.

Gu, C. & Wahba, G. (1993), 'Smoothing spline ANOVA with component-wise Bayesian "confidence intervals"', *J. Computational and Graphical Statistics* **2**, 1–21.

Kimeldorf, G. & Wahba, G. (1970), 'A correspondence between Bayesian estimation of stochastic processes and smoothing by splines', *Ann. Math. Statist.* **41**, 495–502.

Klein, R., Klein, B., Moss, S. Davis, M., & DeMets, D. (1988), Glycosylated hemoglobin predicts the incidence and progression of diabetic retinopathy, *JAMA* **260**, 2864–2871.

O'Sullivan, F., Yandell, B. & Raynor, W. (1986), 'Automatic smoothing of regression functions in generalized linear models', *J. Am. Stat. Soc.* **81**, 96–103.

Wahba, G. (1980), Spline bases, regularization, and generalized cross validation for solving approximation problems with large quantities of noisy data, *in* W. Cheney, ed., 'Approximation Theory III', Academic Press, pp. 905–912.

Wahba, G. (1985), 'A comparison of GCV and GML for choosing the smoothing parameter in the generalized spline smoothing problem', *Ann. Statist.* **13**, 1378–1402.

Wahba, G. (1990), *Spline Models for Observational Data*, SIAM. CBMS-NSF Regional Conference Series in Applied Mathematics, vol. 59.

Wahba, G. (1992), Multivariate function and operator estimation, based on smoothing splines and reproducing kernels, *in* M. Casdagli & S. Eubank, eds, 'Nonlinear Modeling and Forecasting, SFI Studies in the Sciences of Complexity, Proc. Vol XII', Addison-Wesley, pp. 95–112.

Wahba, G., Gu, C., Wang, Y. & Chappell, R. (1993), Soft classification, a. k. a. risk estimation, via penalized log likelihood and smoothing spline analysis of variance, to appear, Proc. Santa Fe Workshop on Supervised Machine Learning, D. Wolpert and A. Lapedes, eds, and Proc. CLNL92, T. Petsche, ed, with permission of all eds.
